# Temporal dynamics of information content carried by neurons in the primary visual cortex

**Danko Nikolić**[*]
Department of Neurophysiology
Max-Planck-Institute for Brain Research,
Frankfurt (Main), Germany
danko@mpih-frankfurt.mpg.de

**Stefan Haeusler**[*]
Institute for Theoretical Computer Science
Graz University of Technology
A-8010 Graz, Austria
haeusler@igi.tugraz.at

**Wolf Singer**
Department of Neurophysiology
Max-Planck-Institute for Brain Research,
Frankfurt (Main), Germany
singer@mpih-frankfurt.mpg.de

**Wolfgang Maass**
Institute for Theoretical Computer Science
Graz University of Technology
A-8010 Graz, Austria
maass@igi.tugraz.at

## Abstract

We use multi-electrode recordings from cat primary visual cortex and investigate whether a simple linear classifier can extract information about the presented stimuli. We find that information is extractable and that it even lasts for several hundred milliseconds after the stimulus has been removed. In a fast sequence of stimulus presentation, information about both new and old stimuli is present simultaneously and nonlinear relations between these stimuli can be extracted. These results suggest nonlinear properties of cortical representations. The important implications of these properties for the nonlinear brain theory are discussed.

## 1   Introduction

It has recently been argued that the most fundamental aspects of computations in visual cortex are still unknown [1]. This could be partially because of the narrow and reductionist approaches in the design of experiments and partially because of the nonlinear properties of cortical neurons that are ignored by the current theories [1]. It has also been argued that the recurrent neuronal circuits in the visual cortex are highly complex and thus, that notions such as "feedforward" and "feedback" are inadequate concepts for the analysis of nonlinear dynamical systems [2]. Furthermore, current theories do not take in account the precise timing of neuronal activity and synchronicity in responses, which should play an important computational role [3].

Alternative computational models from dynamical systems theory [4] argue that fading memory properties of neural circuits are essential for real-time processing of quickly varying visual stimuli. However, an experimental test of this prediction has been missing. An example for an experimental study that may be seen as a step in this direction is [5], where it was shown that the firing activity of neurons in macaque inferior temporal cortex (IT) contains information about an image that has been just presented and that this information lasts for several hundred milliseconds. This information was extracted by algorithms from machine learning that classified the patterns of neuronal responses to different images. The present paper extends the results from [5] in several directions:

---

[*]These authors contributed equally to this work.

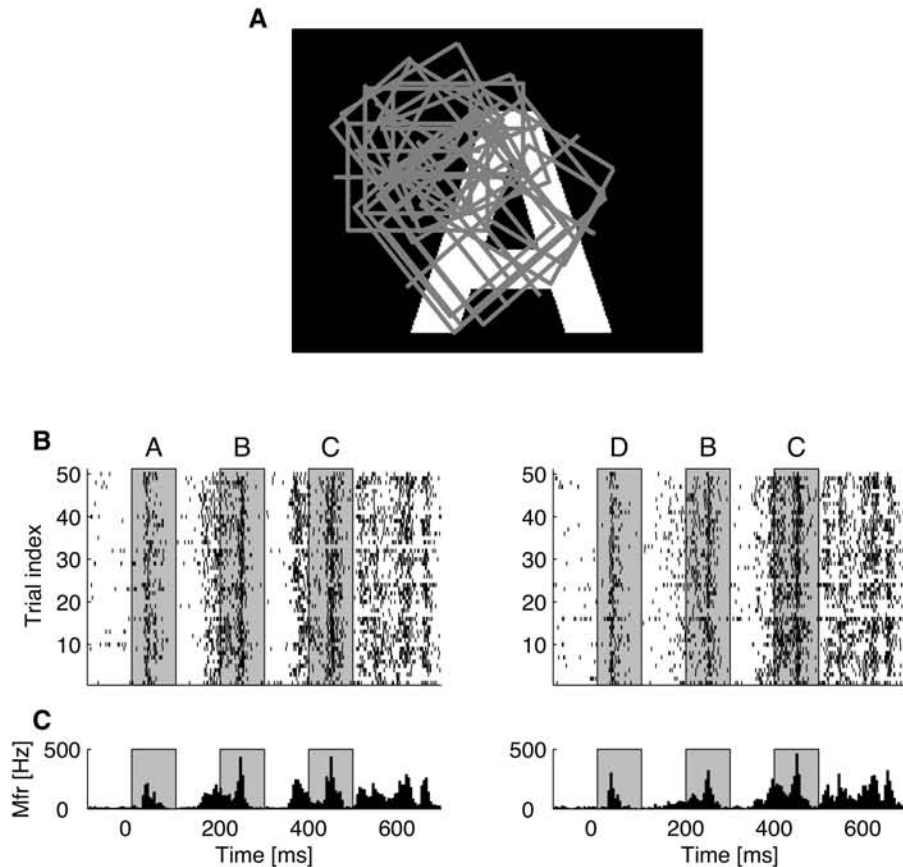

Figure 1: **A:** An example of a visual stimulus in relation to the constellation of receptive fields (gray rectangles) from one Michigan probe. **B:** Spike times recorded from one electrode across 50 stimulus presentations and for two stimulation sequences ($ABC$ and $DBC$). In this and in all other figures the gray boxes indicate the periods during which the letter-stimuli were visible on the screen. **C:** Peri-stimulus time histogram for the responses shown in B. Mfr: mean firing rate.

- We show that also neurons in cat *primary* visual cortex (area 17) and under anesthesia contain information about previously shown images and that this information lasts even longer.
- We analyze the information content in neuronal activity recorded simultaneously from multiple electrodes.
- We analyze the information about a previously shown stimulus for rapid sequences of images and how the information about consecutive images in a sequence is superimposed (i.e., we probe the system's memory for images).

## 2  Methods

### 2.1  Experiments

In three cats anaesthesia was induced with ketamine and maintained with a mixture of 70% $N_2O$ and 30% $O_2$ and with halothane (0.4-0.6%). The cats were paralysed with pancuronium bromide applied intravenously (Pancuronium, Organon, 0.15 mg $kg^{-1}h^{-1}$). Multi-unit activity (MUA) was recorded from area 17 and by using multiple silicon-based 16-channel probes (organized in a $4 \times 4$ spatial matrix) which were supplied by the Center for Neural Communication Technology at the University of Michigan (Michigan probes). The inter-contact distances were 200 $\mu$m (0.3-0.5 M$\Omega$

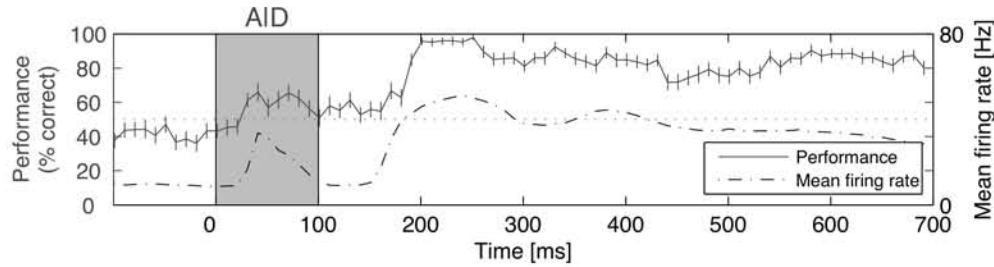

Figure 2: The ability of a linear classifier to determine which of the letters $A$ or $D$ was previously used as a stimulus. The classification performance is shown as a function of time passed between the initiation of the experimental trial and the moment at which a sample of neuronal activity that was taken for training/test of the classifier. The classification performance peaks at about 200 ms (reaching almost 100% accuracy) and remains high until at least 700 ms. Dash-dotted line: the mean firing rate across the entire population of investigated neurons. Dotted line: Performance at the chance level (50% correct).

impedance at 1000 Hz). Signals were amplified 1000× and, to extract unit activity, were filtered between 500 Hz and 3.5 kHz. Digital sampling was made with 32 kHz frequency and the waveforms of threshold-detected action potentials were stored for an off-line spike sorting procedure. The probes were inserted approximately perpendicular to the surface of the cortex, allowing us to record simultaneously from neurons at different cortical layers and at different columns. This setup resulted in a cluster of overlapping receptive fields (RF), all RFs being covered by the stimuli (see Fig. 1A) (more details on recording techniques can be found in [6, 7]).

Stimuli were presented binocularly on a 21" computer monitor (HITACHI CM813ET, 100 Hz refresh rate) and by using the software for visual stimulation ActiveSTIM (www.ActiveSTIM.com). Binocular fusion of the two eyes was achieved by mapping the borders of the respective RFs and then by aligning the optical axes with an adjustable prism placed in front of one eye. The stimuli consisted of single white letters with elementary features suitable for area 17 and spanning approximately 5° of visual angle. The stimuli were presented on a black background for a brief period of time. Fig. 1A illustrates the spatial relation between the constellation of RFs and the stimulus in one of the experimental setups. In each stimulation condition either a single letter was presented or a sequence of up to three letters. For presentation of single letters we used letters A and D each presented for 100 ms.

Stimulus sequences were made with letters A, B, C, D, and E and we compared either the responses across the sequences ABC, DBC, and ADC (cat 1) or across sequences ABE, CBE, ADE, and CDE (cats 2 and 3). Each member of a sequence was presented for 100 ms and the blank delay-period separating the presentation of letters lasted also 100 ms. Each stimulation condition (single letter or a sequence) was presented 50 to 150 times and the order of presentation was randomized across the stimulation conditions. Example raster plots of responses to two different sets of stimuli can be seen in Fig. 1B.

## 2.2 Data analysis

The typical spike trains prior to the application of the spike-sorting procedure are illustrated in Fig. 1B. All datasets showed high trial-to-trial variability, with an average fano factor of about 8. If we included into our analysis all the single units that resulted from the spike-sorting procedure, this resulted in too sparse data representations and hence in overfitting. We therefore used only units with mean firing rates $\geq 10$ Hz and pooled single units with less frequent firings into multi-unit signals. These methods resulted in datasets with 66 to 124 simultaneously recorded units for further analysis.

The recorded spike times were convolved with an exponential kernel with a decay time constant of $\tau = 20$ ms. A linear classifier was trained to discriminate between pairs of stimuli on the basis of the convolved spike trains at time points $t \in \{0, 10,..., 700\}$ ms after stimulus onset (using only the vectors of 66 to 124 values of the convolved time series at time $t$). We refer to this classifier as $R_t$.

A second type of classifier, which we refer to as $R_{int}$, was trained to carry out such classification simultaneously for all time points $t \in \{150, 160,..., 450\}$ (see Fig. 7). If not otherwise stated, the results for type $R_t$ classifiers are reported. A linear classifier applied to the convolved spike data (i.e., an equivalent to low-pass-filtering) can be interpreted as an integrate-and-fire (I&F) neuron with synaptic inputs modeled as Dirac delta functions. The time constant of 20 ms reflects the temporal properties of synaptic receptors and of the membrane. A classification is obtained due to the firing threshold of the I&F neuron.

The classifiers were trained with linear-kernel support vector machines with parameter $C$ chosen to be 10 in case of 50 samples per stimulus class and 50 in case of 150 samples per stimulus. The classification performance was estimated with 10-fold cross validation in which we balanced the number of examples for the training and for the test class. All the reported performance data are for the test class. Error bars in the figures denote the average standard error of the mean for one cross validation run.

## 3 Results

### 3.1 High classification performance

As observed in IT [5], the classification performance peaks also in area 17 at about 200 ms after the stimulus onset. Therefore, a classifier can detect the identity of the stimulus with high reliability. In contrast to [5] information about stimuli is in our data available much longer and can last up to 700 ms after the stimulus onset (Fig. 2).

### 3.2 Memory for the past stimuli

We also find that even when new stimuli are presented, information about old stimuli is not erased. Instead, neuronal activity continued to maintain information about the previously presented stimuli. In Fig. 3 we show that classifiers can extract substantial information about the first image well after this image is removed and when new images are shown. Thus, the system maintains a memory of previous activations and this memory lasts at least several hundred milliseconds. Note that the information remains in memory even if neuronal rate responses decrease for a brief period of time and approach the level that is close to that of spontaneous activity.

### 3.3 Simultaneous availability of different pieces of information

Simultaneous presence of information about different stimuli is a necessary prerequisite for efficient coding. Fig. 4A shows that classifiers can identify the second letter in the sequence. In Fig. 4B we show the results of an experiment in which both the first and second letter were varied simultaneously. Two classifiers, each for the identity of letters at one time slot, performed both very well. During the period from 250 to 300 ms information about both letters was available.

This information can be used to perform a nonlinear XOR classification function, i.e., return one if the sequences $ADE$ or $CBE$ have been presented but not if both A and B or none of them was presented in the same sequence, in which case a zero should be returned. In Fig. 4C we show XOR classification based on the information extracted from the two classifiers in Fig. 4B (dashed line). In this case, the nonlinear component of the XOR computation is made externally by the observer and not by the brain. We compared these results with the performance of a single linear classifier that was trained to extract XOR information directly from the brain responses (solid line). As this classifier was linear, the nonlinear component of the computation could have been performed only by the brain.

The classification performance was in both cases well above the chance level (horizontal dotted line in Fig. 4C). More interestingly, the two performance functions were similar, the brain slightly outperforming the external computation of XOR in this nonlinear task. Therefore, the brain can perform also nonlinear computations.

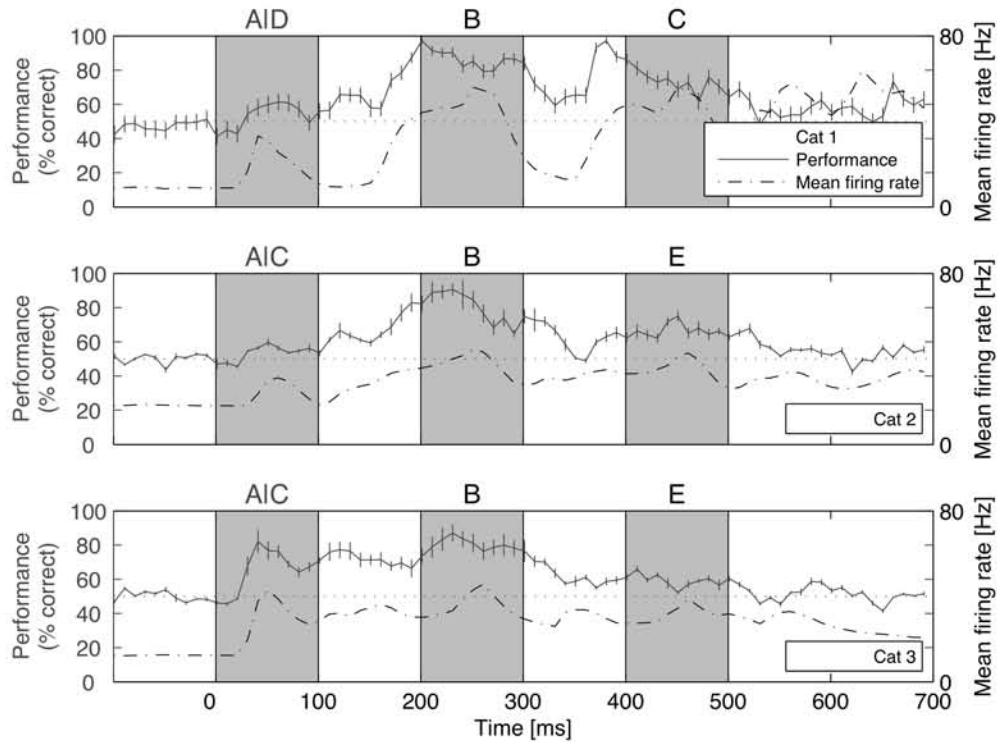

Figure 3: Classifiers were trained to identify the first letter in the sequences $ABC$ vs. $DBC$ in one experiment (cat 1) and sequences $ABE$ vs. $CBE$ in other two experiments (cats 2 and 3). In all cases, the performance reached its maximum shortly before or during the presentation of the second letter. In one case (cat 1) information about the first letter remained present even with multiple exchanges of the stimuli, i.e., during the presentation of the third letter. Notification is the same as in Fig. 2.

## 3.4 Neural code

It is also important to understand how this information is encoded in neuronal activity. Fig. 5 shows lower bounds for the information contents of neuronal firing rates. The ability of the classifiers to distinguish between two stimuli was positively correlated to the difference in the average firing rate responses to these stimuli. For the three experiments (cats 1 to 3) the Pearson coefficients of correlation between these two variables were 0.37, 0.42 and 0.46, respectively (14-21% of explained variance). The correlation coefficients with the absolute rate responses were always larger (0.45, 0.68 and 0.66).

In contrast to [5], we also found that in addition to rate responses, the precise timing relationships between neuronal spiking events carry important information about stimulus identity. To show this, we perturbed the recorded data by jittering the timings of the spikes for various amounts. Only a few milliseconds of jitter were sufficient to decrease the performance of the classifiers significantly (Fig. 6). Therefore, the information is also contained in the timing of spikes. Timing is therefore also a neuronal code. Moreover, like rate, timing also carried information about the past. We could demonstrate that jitter induces a significant drop in classification performance even for time points as far as 200 ms past the stimulus onset (the rightmost panel of Fig. 6).

We also investigated the 'synaptic' weights of the classifiers and this enabled us to study the temporal evolution of the code. We asked the following question: Do the same pieces of information indicate the identity of a stimulus early and late in the stimulation sequence? Hence, we compared the performance of $R_t$ classifiers, for which the weights were allowed to change along the stimulation sequence, against the performance of $R_{int}$ classifiers, for which the weights were fixed. The results indicated that the neuronal code was invariant during a single stimulation-response event (e.g., on-

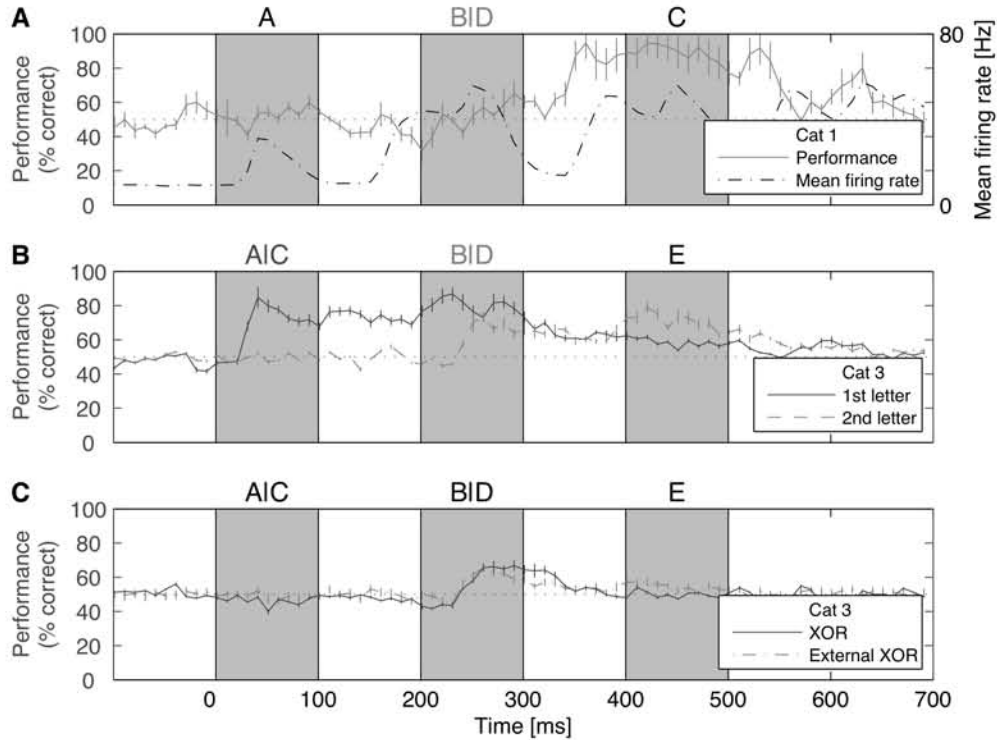

Figure 4: Classification of the second letter in a sequence and of a combination of letters. **A:** Performance of a classifier trained to identify the second letter in the sequences $ABC$ and $ADC$. Similarly to the results in Fig. 3, the performance is high during the presentation of the subsequent letter. **B:** Simultaneously available information about two different letters of a sequence. Two classifiers identified either the first or the second letter of the following four sequences: $ABE$, $CBE$, $ADE$, and $CDE$. **C:** The same data as in B but a linear classifier was trained to compute the XOR function of the 2 bits encoded by the 2 choices $A|C$ and $B|D$ (solid line). The dashed line indicates the performance of a control calculation made by an external computation of the XOR function that was based on the information extracted by the classifiers whose performance functions are plotted in B.

responses to the presentation of a letter) but changed across such events (e.g., off-response to the same letter or on-response to the subsequent letter)(Fig. 7).

Finally, as in [5], an application of nonlinear radial-basis kernels did not produce significant improvement in the number of correct classifications when compared to linear kernels and this was the case for type $R_t$ classifiers for which the improvement never exceeded 2% (results not shown). However, the performance of type $R_{int}$ classifiers increased considerably ($\approx$8%) when they were trained with nonlinear as opposed to linear kernels (time interval $t$ =[150, 450] ms, results not shown).

## 4 Discussion

In the present study we find that information about preceding visual stimuli is preserved for several hundred ms in neurons of the primary visual cortex of anesthetized cats. These results are consistent to those reported by [5] who investigated neuronal activity in awake state and in a higher cortical area (IT-cortex). We show that information about a previously shown stimulus can last in visual cortex up to 700 ms, much longer than reported for IT-cortex. Hence, we can conclude that it is a general property of cortical networks to contain information about the stimuli in a distributed and time-dynamic manner. Thus, a trained classifier is capable to reliably determine from a short sample of this distributed activity the identity of previously presented stimuli.

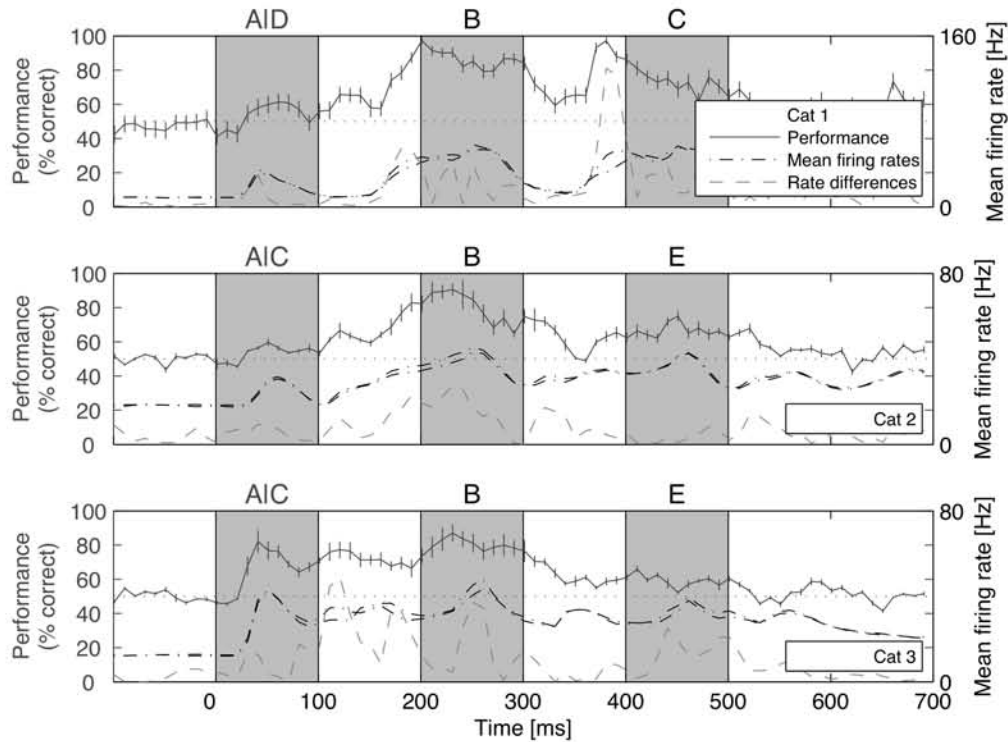

Figure 5: The relation between classifier's performance and i) the mean firing rates (dash-dotted lines) and ii) the difference in the mean firing rates between two stimulation conditions (8 fold magnified, dashed lines). The results are for the same data as in Fig. 3.

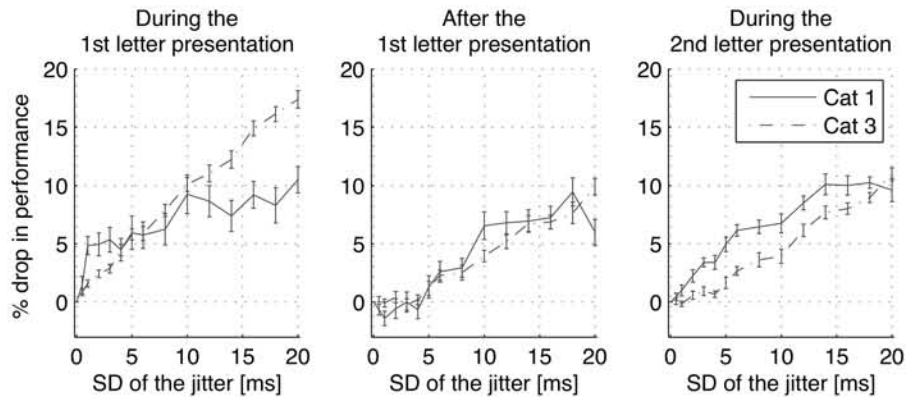

Figure 6: Drop in performance for the classifiers in Fig. 3 due to the Gaussian jitter of spiking times. The drop in performance was computed for three points in time according to the original peaks in performance in Fig. 3. For cat 1, these peaks were $t \in \{60, 120, 200\}$ ms and for cat 3, $t \in \{40, 120, 230\}$ ms. The performance drops for these three points in time are shown in the three panels, respectively and in the order left to right. SD: standard deviation of jitter. A standard deviation of only a few milliseconds decreased the classification performance significantly.

Furthermore, the system's memory for the past stimulation is not necessarily erased by the presentation of a new stimulus, but it is instead possible to extract information about multiple stimuli simultaneously. We show that different pieces of information are superimposed to each other and that they allow extraction of nonlinear relations between the stimuli such as the XOR function.

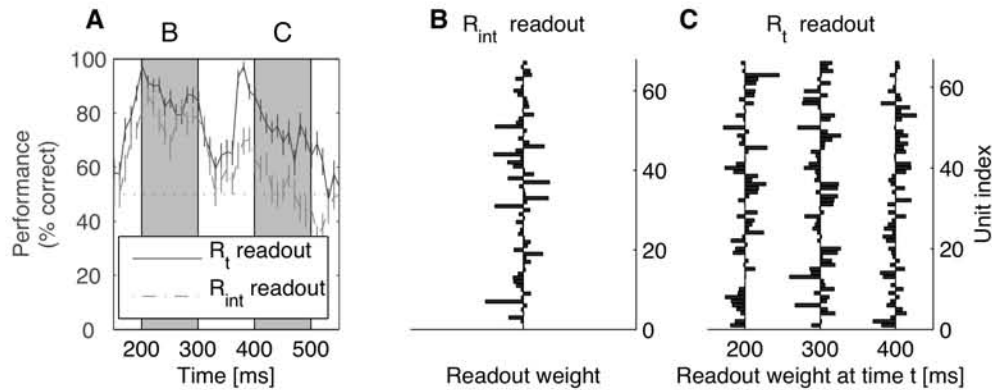

Figure 7: Temporal evolution of the weights needed for optimal linear classification. **A:** Comparison in performance between $R_t$ and $R_{int}$ classifiers. $R_{int}$ classifier was trained on the time intervals $t =$ [150, 450] ms and on the data in Fig. 3. The performance drop of a type $R_{int}$ classifier during the presentation of the third letter indicates that the neural code has changed since the presentation of the second letter. **B:** Weight vector of the type $R_{int}$ classifier used in A. **C:** Weight vectors of the type $R_t$ classifier shown in A for $t \in \{200, 300, 400\}$ ms.

Our results indicate that the neuronal code is not only contained in rate responses but that the precise spike-timing relations matter as well and that they carry additional and important information about the stimulus. Furthermore, almost all information extracted by the state-of-the-art nonlinear classifiers can be extracted by using simple linear classification mechanisms. This is in agreement with the results reported in [5] for IT-cortex. Hence, similarly to our classifiers, cortical neurons should also be able to read out such information from distributed neuronal activity.

These results have important implications for theories of brain function and for understanding the nature of computations performed by natural neuronal circuits. In agreement with the recent criticism [1, 2], the present results are not compatible with computational models that require a precise "frame by frame" processing of visual inputs or focus on comparing each frame with an internally generated reconstruction or prediction. These models require a more precise temporal organisation of information about subsequent frames of visual inputs. Instead, our results support the view recently put forward by theoretical studies [4, 8], in which computations are performed by complex dynamical systems while information about results of these computations is read out by simple linear classifiers. These theoretical systems show memory and information-superposition properties that are similar to those reported here for the cerebral cortex.

# References

[1] B. A. Olshausen and D. J. Field. What is the other 85% of v1 doing? In J. L. van Hemmen and T. J. Sejnowski, editors, *23 Problems in Systems Neuroscience*, pages 182–211. Oxford Univ. Press (Oxford, UK), 2006.

[2] A. M. Sillito and H. E. Jones. Feedback systems in visual processing. In L. M. Chalupa and J. S. Werner, editors, *The Visual Neurosciences*, pages 609–624. MIT Press, 2004.

[3] W. Singer. Neuronal synchrony: a versatile code for the definition of relations? *Neuron*, 24(1):49–65, 111–25, 1999.

[4] W. Maass, T. Natschläger, and H. Markram. Real-time computing without stable states: A new framework for neural computation based on perturbations. *Neural Computation*, 14(11):2531–2560, 2002.

[5] C. P. Hung, G. Kreiman, T. Poggio, and J. J. DiCarlo. Fast readout of object identity from macaque inferior temporal cortex. *Science*, 310(5749):863–866, 2005.

[6] G. Schneider, M. N. Havenith, and D. Nikolić. Spatio-temporal structure in large neuronal networks detected from cross correlation. *Neural Computation*, 18(10):2387–2413, 2006.

[7] G. Schneider and D. Nikolić. Detection and assessment of near-zero delays in neuronal spiking activity. *J Neurosci Methods*, 152(1-2):97–106, 2006.

[8] H. Jäger and H. Haas. Harnessing nonlinearity: predicting chaotic systems and saving energy in wireless communication. *Science*, 304:78–80, 2004.
